# Nested sampling for Potts models

**Iain Murray**
Gatsby Computational Neuroscience Unit
University College London
i.murray@gatsby.ucl.ac.uk

**David J.C. MacKay**
Cavendish Laboratory
University of Cambridge
mackay@mrao.cam.ac.uk

**Zoubin Ghahramani**
Gatsby Computational Neuroscience Unit
University College London
zoubin@gatsby.ucl.ac.uk

**John Skilling**
Maximum Entropy
Data Consultants Ltd.
skilling@eircom.net

## Abstract

Nested sampling is a new Monte Carlo method by Skilling [1] intended for general Bayesian computation. Nested sampling provides a robust alternative to annealing-based methods for computing normalizing constants. It can also generate estimates of other quantities such as posterior expectations. The key technical requirement is an ability to draw samples uniformly from the prior subject to a constraint on the likelihood. We provide a demonstration with the Potts model, an undirected graphical model.

## 1 Introduction

The computation of normalizing constants plays an important role in statistical inference. For example, Bayesian model comparison needs the evidence, or marginal likelihood of a model $\mathcal{M}$

$$\mathcal{Z} = p(\mathcal{D}|\mathcal{M}) = \int p(\mathcal{D}|\theta, \mathcal{M})p(\theta|\mathcal{M})\,\mathrm{d}\theta \equiv \int L(\theta)\pi(\theta)\,\mathrm{d}\theta, \qquad (1)$$

where the model has prior $\pi$ and likelihood $L$ over parameters $\theta$ after observing data $\mathcal{D}$. This integral is usually intractable for models of interest. However, given its importance in Bayesian model comparison, many approaches—both sampling-based and deterministic—have been proposed for estimating it.

Often the evidence cannot be obtained using samples drawn from either the prior $\pi$, or the posterior $p(\theta|\mathcal{D}, \mathcal{M}) \propto L(\theta)\pi(\theta)$. Practical Monte Carlo methods need to sample from a sequence of distributions, possibly at different "temperatures" $p(\theta|\beta) \propto L(\theta)^\beta \pi(\theta)$ (see Gelman and Meng [2] for a review). These methods are sometimes cited as a gold standard for comparison with other approximate techniques, e.g. Beal and Ghahramani [3]. However, care is required in choosing intermediate distributions; appropriate temperature-based distributions may be difficult or impossible to find. Nested sampling provides an alternate standard, which makes no use of temperature and does not require tuning of intermediate distributions or other large sets of parameters.

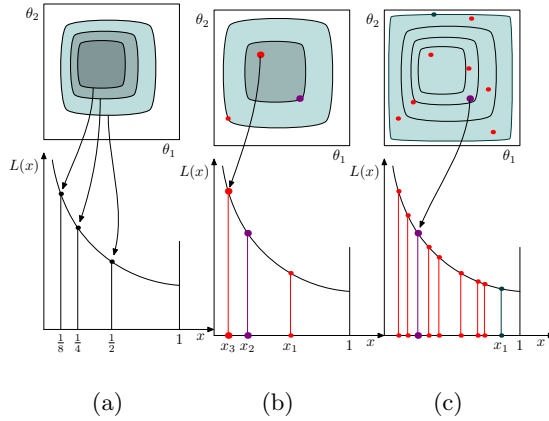

Figure 1: (a) Elements of parameter space (top) are sorted by likelihood and arranged on the $x$-axis. An eighth of the prior mass is inside the innermost likelihood contour in this figure. (b) Point $x_i$ is drawn from the prior inside the likelihood contour defined by $x_{i-1}$. $L_i$ is identified and $p(\{x_i\})$ is known, but exact values of $x_i$ are not known. (c) With $N$ particles, the least likely one sets the likelihood contour and is replaced by a new point inside the contour ($\{L_i\}$ and $p(\{x_i\})$ are still known).

(a)        (b)        (c)

Nested sampling uses a natural definition of $\mathcal{Z}$, a sum over prior mass. The weighted sum over likelihood elements is expressed as the area under a monotonic one-dimensional curve "$L$ vs $x$" (figure 1(a)), where:

$$\mathcal{Z} = \int L(\theta)\pi(\theta) \, \mathrm{d}\theta = \int_0^1 L(\theta(x)) \, \mathrm{d}x. \tag{2}$$

This is a change of variables $\mathrm{d}x(\theta) = \pi(\theta)\mathrm{d}\theta$, where each volume element of the prior in the original $\theta$-vector space is mapped onto a scalar element on the one-dimensional $x$-axis. The ordering of the elements on the $x$-axis is chosen to sort the prior mass in decreasing order of likelihood values ($x_1 < x_2 \Rightarrow L(\theta(x_1)) > L(\theta(x_2))$). See appendix A for dealing with elements with identical likelihoods.

Given some points $\{(x_i, L_i)\}_{i=1}^I$ ordered such that $x_i > x_{i+1}$, the area under the curve (2) is easily approximated. We denote by $\hat{\mathcal{Z}}$ estimates obtained using a trapezoidal rule. Rectangle rules upper and lower bound the error $\hat{\mathcal{Z}} - \mathcal{Z}$.

Points with known $x$-coordinates are unavailable in general. Instead we generate points, $\{\theta_i\}$, such that the *distribution* $p(\mathbf{x})$ is known (where $\mathbf{x} \equiv \{x_i\}$), and find their associated $\{L_i\}$. A simple algorithm to draw $I$ points is algorithm 1, see also figure 1(b).

| **Algorithm 1** | **Algorithm 2** |
|---|---|
| **Initial point:** draw $\theta_1 \sim \pi(\theta)$. | **Initialize:** draw $N$ points $\theta^{(n)} \sim \pi(\theta)$ |
| **for i = 2 to I:** draw $\theta_i \sim \breve{\pi}(\theta\|L(\theta_{i-1}))$, | **for i = 2 to I:** |
| where |   • $m = \mathrm{argmin}_n L(\theta^{(n)})$ |
| $$\breve{\pi}(\theta\|L(\theta_{i-1})) \propto \begin{cases} \pi(\theta) & L(\theta) > L(\theta_{i-1}) \\ 0 & \text{otherwise.} \end{cases} \tag{3}$$ |   • $\theta_{i-1} = \theta^{(m)}$ |
| |   • draw $\theta_m \sim \breve{\pi}(\theta\|L(\theta_{i-1}))$, given by equation (3) |

We know $p(x_1) = \text{Uniform}(0, 1)$, because $x$ is a cumulative sum of prior mass. Similarly $p(x_i|x_{i-1}) = \text{Uniform}(0, x_{i-1})$, as every point is drawn from the prior subject to $L(\theta_i) > L(\theta_{i-1}) \Rightarrow x_i < x_{i-1}$. This recursive relation allows us to compute $p(\mathbf{x})$.

A simple generalization, algorithm 2, uses multiple $\theta$ particles; at each step the least likely is replaced with a draw from a constrained prior (figure 1(c)). Now $p(x_1|N) = Nx_1^{N-1}$ and subsequent points have $p(x_i/x_{i-1}|x_{i-1}, N) = N(x_i/x_{i-1})^{N-1}$. This

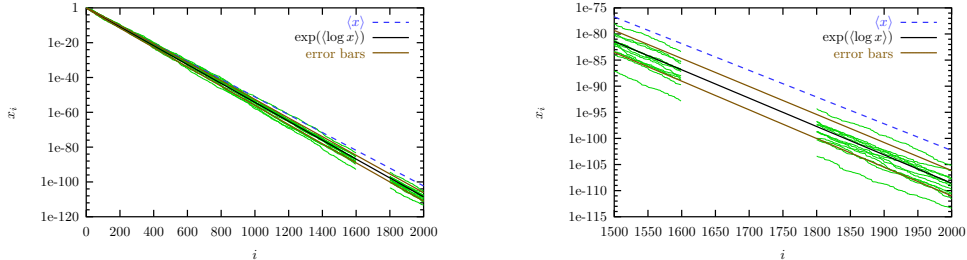

Figure 2: The arithmetic and geometric means of $x_i$ against iteration number, $i$, for algorithm 2 with $N = 8$. Error bars on the geometric mean show $\exp(-i/N \pm \sqrt{i}/N)$. Samples of $p(\mathbf{x}|N)$ are superimposed ($i = 1600 \ldots 1800$ omitted for clarity).

distribution over $\mathbf{x}$ combined with observations $\{L_i\}$ gives a distribution over $\hat{\mathcal{Z}}$:

$$p(\hat{\mathcal{Z}}|\{L_i\}, N) \approx \int \delta(\hat{\mathcal{Z}}(\mathbf{x}) - \hat{\mathcal{Z}}) p(\mathbf{x}|N) \; \mathrm{d}\mathbf{x}. \tag{4}$$

Samples from the posterior over $\theta$ are also available, see Skilling [1] for details.

Nested sampling was introduced by Skilling [1]. The key idea is that samples from the prior, subject to a nested sequence of constraints (3), give a probabilistic realization of the curve, figure 1(a). Related work can be found in McDonald and Singer [4]. Explanatory notes and some code are available online[1]. In this paper we present some new discussion of important issues regarding the practical implementation of nested sampling and provide the first application to a challenging problem. This leads to the first cluster-based method for Potts models with first-order phase transitions of which we are aware.

## 2   Implementation issues

### 2.1   MCMC approximations

The nested sampling algorithm assumes obtaining samples from $\breve{\pi}(\theta|L(\theta_{i-1}))$, equation (3), is possible. Rejection sampling using $\pi$ would slow down exponentially with iteration number $i$. We explore approximate sampling from $\breve{\pi}$ using Markov chain Monte Carlo (MCMC) methods.

In high-dimensional problems it is likely that the majority of $\breve{\pi}$'s mass is typically in a thin shell at the contour surface [5, p37]. This suggests finding efficient chains that sample at constant likelihood, a *microcanonical* distribution. In order to complete an ergodic MCMC method, we also need transition operators that can alter the likelihood (within the constraint). A simple Metropolis method may suffice.

We must initialize the Markov chain for each new sample somewhere. One possibility is to start at the position of the deleted point, $\theta_{i-1}$, on the contour constraint, which is independent of the other points and not far from the bulk of the required uniform distribution. However, if the Markov chain mixes slowly amongst modes, the new point starting at $\theta_{i-1}$ may be trapped in an insignificant mode. In this case it would be better to start at one of the other $N-1$ existing points inside the contour constraint. They are all draws from the correct distribution, $\breve{\pi}(\theta|L(\theta_{i-1}))$, so represent modes fairly. However, this method may also require many Markov chain steps, this time to make the new point effectively independent of the point it cloned.

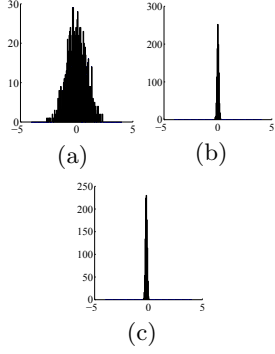

(a)             (b)

(c)

Figure 3: Histograms of errors in the point estimate $\log(\tilde{\mathcal{Z}})$ over 1000 random experiments for different approximations. The test system was a 40-dimensional hypercube of length 100 with uniform prior centered on the origin. The log-likelihood was $L = -\theta^\top\theta/2$. Nested sampling used $N = 10$, $I = 2000$. (a) Monte Carlo estimation (equation (5)) using $S = 12$ sampled trajectories (b) $S = 1200$ sampled trajectories. (c) Deterministic approximation using the geometric mean trajectory. In this example perfect integration over $p(\mathbf{x}|N)$ gives a distribution of width $\approx 3$ over $\log(\hat{\mathcal{Z}})$. Therefore, improvements over (c) for approximating equation (5) are unwarranted.

## 2.2 Integrating out x

To estimate quantities of interest, we average over $p(\mathbf{x}|N)$, as in equation (4). The mean of a distribution over $\log(\hat{\mathcal{Z}})$ can be found by simple Monte Carlo estimation:

$$\log(\mathcal{Z}) \approx \int \log(\hat{\mathcal{Z}}(\mathbf{x})) p(\mathbf{x}|N) \, \mathrm{d}\mathbf{x} \approx \frac{1}{S}\sum_{s=1}^{S} \log(\hat{\mathcal{Z}}(\mathbf{x}^{(s)})) \qquad \mathbf{x}^{(s)} \sim p(\mathbf{x}|N). \quad (5)$$

This scheme is easily implemented for any expectation under $p(\mathbf{x}|N)$, including error bars from the variance of $\log(\hat{\mathcal{Z}})$. To reduce noise in comparisons between runs it is advisable to reuse the same samples from $p(\mathbf{x}|N)$ (e.g. clamp the seed used to generate them).

A simple deterministic approximation is useful for understanding, and also provides fast to compute, low variance estimators. Figure 2 shows sampled trajectories of $x_i$ as the algorithm progresses. The geometric mean path, $x_i \approx \exp(\int p(x_i|N)\log x_i \, \mathrm{d}x_i) = e^{-i/N}$, follows the path of typical settings of $\mathbf{x}$. Using this single $\mathbf{x}$ setting is a reasonable and very cheap alternative to averaging over settings (equation 5); see figure 3.

Typically the trapezoidal estimate of the integral, $\hat{\mathcal{Z}}$, is dominated by a small number of trapezoids, around iteration $i^*$ say. Considering uncertainty on just $\log x_{i^*} = -i^*/N \pm \sqrt{i^*}/N$ provides reasonable and convenient error bars.

## 3 Potts Models

The Potts model, an undirected graphical model, defines a probability distribution over discrete variables $\mathbf{s} = (s_1, \ldots, s_n)$, each taking on one of $q$ distinct "colors":

$$P(\mathbf{s}|J, q) = \frac{1}{\mathcal{Z}_\mathrm{P}(J, q)} \exp\left( \sum_{(ij)\in\mathcal{E}} J(\delta_{s_i s_j} - 1) \right). \quad (6)$$

The variables exist as nodes on a graph where $(ij) \in \mathcal{E}$ means that nodes $i$ and $j$ are linked by an edge. The Kronecker delta, $\delta_{s_i s_j}$ is one when $s_i$ and $s_j$ are the same color and zero otherwise. Neighboring nodes pay an "energy penalty" of $J$ when they are different colors. Here we assume identical positive couplings $J > 0$ on each edge (section 4 discusses the extension to different $J_{ij}$). The Ising model and Boltzmann machine are both special cases of the Potts model with $q=2$.

Our goal is to compute the normalization constant $\mathcal{Z}_\mathrm{P}(J, q)$, where the discrete variables $\mathbf{s}$ are the $\theta$ variables that need to be integrated (i.e. summed) over.

## 3.1 Swendsen–Wang sampling

We will take advantage of the "Fortuin-Kasteleyn-Swendsen-Wang" (FKSW) joint distribution identified explicitly in Edwards and Sokal [6] over color variables $\mathbf{s}$ and a bond variable for each edge in $\mathcal{E}$, $d_{ij} \in \{0,1\}$:

$$P\left(\mathbf{s}, \mathbf{d}\right) = \frac{1}{\mathcal{Z}_{\mathrm{P}}(J,q)} \prod_{(ij) \in \mathcal{E}} \left[ (1-p)\delta_{d_{ij},0} + p\delta_{d_{ij},1}\delta_{s_i,s_j} \right], \qquad p \equiv (1 - e^{-J}). \quad (7)$$

The marginal distribution over $\mathbf{s}$ in the FKSW model is the Potts distribution, equation (6). The marginal distribution over the bonds is the random cluster model of Fortuin and Kasteleyn [7]:

$$P\left(\mathbf{d}\right) = \frac{1}{\mathcal{Z}_{\mathrm{P}}(J,q)} p^D (1-p)^{|\mathcal{E}|-D} q^{C(\mathbf{d})} = \frac{1}{\mathcal{Z}_{\mathrm{P}}(J,q)} \exp(D \log(e^J - 1)) e^{-J|\mathcal{E}|} q^{C(\mathbf{d})}, \quad (8)$$

where $C(\mathbf{d})$ is the number of connected components in a graph with edges wherever $d_{ij} = 1$, and $D = \sum_{(ij) \in \mathcal{E}} d_{ij}$. As the partition functions of equations 6, 7 and 8 are identical, we should consider using any of these distributions to compute $\mathcal{Z}_{\mathrm{P}}(J,q)$. The algorithm of Swendsen and Wang [8] performs block Gibbs sampling on the joint model by alternately sampling from $P(d_{ij}|\mathbf{s})$ and $P(\mathbf{s}|d_{ij})$. This can convert a sample from any of the three distributions into a sample from one of the others.

## 3.2 Nested Sampling

A simple approximate nested sampler uses a fixed number of Gibbs sampling updates of $\breve{\pi}$. Cluster-based updates are also desirable in these models. Focusing on the random cluster model, we rewrite equation (8):

$$P\left(\mathbf{d}\right) = \frac{1}{\mathcal{Z}_{\mathrm{N}}} L(\mathbf{d})\pi(\mathbf{d}) \quad \text{where} \quad (9)$$

$$\mathcal{Z}_{\mathrm{N}} = \frac{\mathcal{Z}_{\mathrm{P}}(J,q)}{\mathcal{Z}_\pi} \exp(J|\mathcal{E}|), \quad L(\mathbf{d}) = \exp(D \log(e^J - 1)), \quad \pi(\mathbf{d}) = \frac{1}{\mathcal{Z}_\pi} q^{C(\mathbf{d})}.$$

Likelihood thresholds are thresholds on the total number of bonds $D$. Many states have identical $D$, which requires careful treatment, see appendix A. Nested sampling on this system will give the ratio of $\mathcal{Z}_{\mathrm{P}}/\mathcal{Z}_\pi$. The prior normalization, $\mathcal{Z}_\pi$, can be found from the partition function of a Potts system at $J = \log(2)$.

The following steps give two MCMC operators to change the bonds $\mathbf{d} \to \mathbf{d}'$:

1. Create a random coloring, $\mathbf{s}$, uniformly from the $q^{C(\mathbf{d})}$ colorings satisfying the bond constraints $\mathbf{d}$, as in the Swendsen–Wang algorithm.
2. Count sites that allow bonds, $E = \sum_{(ij) \in \mathcal{E}} \delta_{s_i,s_j}$.
3. Either, operator 1: record the number of bonds $D' = \sum_{(ij) \in \mathcal{E}} d_{ij}$

   Or, operator 2: draw $D'$ from $Q(D'|E(\mathbf{s})) \propto \binom{E(\mathbf{s})}{D'}$.
4. Throw away the old bonds, $\mathbf{d}$, and pick uniformly from one of the $\binom{E(\mathbf{s})}{D'}$ ways of setting $D'$ bonds in the $E$ available sites.

The probability of proposing a particular coloring and new setting of the bonds is

$$Q(\mathbf{s}, \mathbf{d}'|\mathbf{d}) = Q(\mathbf{d}'|\mathbf{s}, D')Q(D'|E(\mathbf{s}))Q(\mathbf{s}|\mathbf{d}) = \frac{1}{\binom{E(\mathbf{s})}{D'}} Q(D'|E(\mathbf{s})) \frac{1}{q^{C(\mathbf{d})}}. \quad (10)$$

Summing over colorings, the correct Metropolis-Hastings acceptance ratio is:

$$a = \frac{\pi(\mathbf{d}')}{\pi(\mathbf{d})} \cdot \frac{\sum_{\mathbf{s}} Q(\mathbf{s}, \mathbf{d}|\mathbf{d}')}{\sum_{\mathbf{s}} Q(\mathbf{s}, \mathbf{d}'|\mathbf{d})} = \frac{q^{C(\mathbf{d}')}}{q^{C(\mathbf{d})}} \cdot \frac{q^{C(\mathbf{d})}}{q^{C(\mathbf{d}')}} \frac{\sum_{\mathbf{s}} Q(D|\mathbf{s})/\binom{E(\mathbf{s})}{D}}{\sum_{\mathbf{s}} Q(D'|\mathbf{s})/\binom{E(\mathbf{s})}{D'}} = 1, \quad (11)$$

Table 1: Partition function results for $16\times16$ Potts systems (see text for details).

| Method | $q = 2$ (Ising), $J = 1$ | $q = 10$, $J = 1.477$ |
|---|---|---|
| Gibbs AIS | $7.1 \pm 1.1$ | $(1.5)$ |
| Swendsen–Wang AIS | $7.4 \pm 0.1$ | $(1.2)$ |
| Gibbs nested sampling | $7.1 \pm 1.0$ | $12.2 \pm 2.4$ |
| Random-cluster nested sampling | $7.1 \pm 0.7$ | $14.1 \pm 1.8$ |
| Acceptance ratio | $7.3$ | $11.2$ |

regardless of the choice in step 3. The simple first choice solves the difficult problem of navigating at constant $D$. The second choice defines an ergodic chain[2].

## 4   Results

Table 1 shows results on two example systems: an Ising model, $q = 1$, and a $q = 10$ Potts model in an difficult parameter regime. We tested nested samplers using Gibbs sampling and the cluster-based algorithm, annealed importance sampling (AIS) [9] using both Gibbs sampling and Swendsen–Wang cluster updates. We also developed an acceptance ratio method [10] based on our representation in equation (9), which we ran extensively and should give nearly correct results.

Annealed importance sampling (AIS) was run 100 times, with a geometric spacing of $10^4$ settings of $J$ as the annealing schedule. Nested sampling used $N = 100$ particles and 100 full-system MCMC updates to approximate each draw from $\breve{\pi}$. Each Markov chain was initialized at one of the $N-1$ particles satisfying the current constraint. In trials using the other alternative (section 2.1) the Gibbs nested sampler could get stuck permanently in a local maximum of the likelihood, while the cluster method gave erroneous answers for the Ising system.

AIS performed very well on the Ising system. We took advantage of its performance in easy parameter regimes to compute $\mathcal{Z}_\pi$ for use in the cluster-based nested sampler. However, with a "temperature-based" annealing schedule, AIS was unable to give useful answers for the $q = 10$ system. While nested sampling appears to be correct within its error bars.

It is known that even the efficient Swendsen–Wang algorithm mixes slowly for Potts models with $q > 4$ near critical values of $J$ [11], see figure 4. Typical Potts model states are either entirely disordered or ordered; disordered states contain a jumble of small regions with different colors (e.g. figure 4(b)), in ordered states the system is predominantly one color (e.g. figure 4(d)). Moving between these two phases is difficult; defining a valid MCMC method that moves between distinct phases requires knowledge of the relative probability of the whole collections of states in those phases.

Temperature-based annealing algorithms explore the model for a range of settings of $J$ and fail to capture the correct behavior near the transition. Despite using closely related Markov chains to those used in AIS, nested sampling can work in all parameter regimes. Figure 4(e) shows how nested sampling can explore a mixture of ordered and disordered phases. By moving steadily through these states, nested sampling is able to estimate the prior mass associated with each likelihood value.

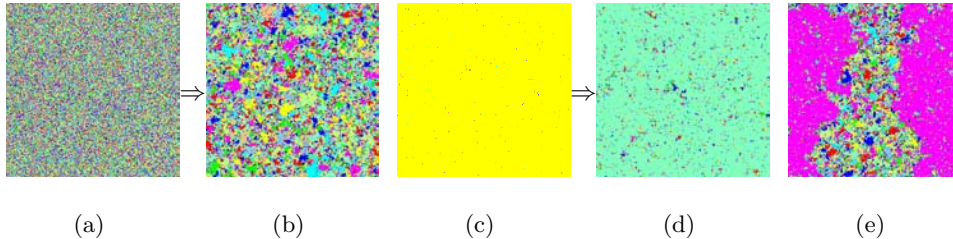

|  (a)  |  (b)  |  (c)  |  (d)  |  (e)  |

Figure 4: Two $256 \times 256$, $q = 10$ Potts models with starting states (a) and (c) were simulated with $5 \times 10^6$ full-system Swendsen–Wang updates with $J = 1.42577$. The corresponding results, (b) and (d) are typical of all the intermediate samples: Swendsen–Wang is unable to take (a) into an ordered phase, or (c) into a disordered phase, although both phases are typical at this $J$. (e) in contrast shows an intermediate state of nested sampling, which succeeds in bridging the phases.

This behaviour is not possible in algorithms that use $J$ as a control parameter.

The potentials on every edge of the Potts model in this paper were the same. Much of the formalism above generalizes to allow different edge weights $J_{ij}$ on each edge, and non-zero biases on each variable. Indeed Edwards and Sokal [6] gave a general procedure for constructing such auxiliary-variable joint distributions. This generalization would make the model more relevant to MRFs used in other fields (e.g. computer vision). The challenge for nested sampling remains the invention of effective sampling schemes that keep a system at or near constant energy. Generalizing step 4 in section 3.2 would be the difficult step.

Other temperatureless Monte Carlo methods exist, e.g. Berg and Neuhaus [12] study the Potts model using the *multicanonical ensemble*. Nested sampling has some unique properties compared to the established method. Formally it has only one free parameter, $N$ the number of particles. Unless problems with multiple modes demand otherwise, $N = 1$ often reveals useful information, and if the error bars on $\mathcal{Z}$ are too large further runs with larger $N$ may be performed.

## 5  Conclusions

We have applied nested sampling to compute the normalizing constant of a system that is challenging for many Monte Carlo methods.

- Nested sampling's key technical requirement, an ability to draw samples uniformly from a constrained prior, is largely solved by efficient MCMC methods.

- No complex schedules are required; steady progress towards compact regions of large likelihood is controlled by a single free parameter, $N$, the number of particles.

- Multiple particles, a built-in feature of this algorithm, are often necessary to obtain accurate results.

- Nested sampling has no special difficulties on systems with first order phase-transitions, whereas all temperature-based methods fail.

We believe that nested sampling's unique properties will be found useful in a variety of statistical applications.

# A    Degenerate likelihoods

The description in section 1 assumed that the likelihood function provides a total ordering of elements of the parameter space. However, distinct elements $dx$ and $dx'$ could have the same likelihood, either because the parameters are discrete, or because the likelihood is degenerate.

One way to break degeneracies is through a joint model with variables of interest $\theta$ and an independent variable $m \in [0, 1]$:

$$P(\theta, m) = P(\theta) \cdot P(m) = \frac{1}{\mathcal{Z}} L(\theta)\pi(\theta) \cdot \frac{1}{\mathcal{Z}_m} L(m)\pi(m) \tag{12}$$

where $L(m) = 1 + \epsilon(m - 0.5)$, $\pi(m) = 1$ and $\mathcal{Z}_m = 1$. We choose $\epsilon$ such that $\log(\epsilon)$ is smaller than the smallest difference in $\log(L(\theta))$ allowed by machine precision. Standard nested sampling is now possible. Assuming we have a likelihood constraint $L_i$, we need to be able to draw from

$$P(\theta', m' | \theta, m, L_i) \propto \begin{cases} \pi(\theta')\pi(m') & L(\theta')L(m') > L_i, \\ 0 & \text{otherwise.} \end{cases} \tag{13}$$

The additional variable can be ignored except for $L(\theta') = L(\theta_i)$, then only $m' > m$ are possible. Therefore, the probability of states with likelihood $L(\theta_i)$ are weighted by $(1 - m')$.

## Footnotes

[1]`http://www.inference.phy.cam.ac.uk/bayesys/`

[2]Proof: with finite probability all $s_i$ are given the same color, then any allowable $D'$ is possible, in turn all allowable $\mathbf{d}'$ have finite probability.

# References

[1] John Skilling. Nested sampling. In R. Fischer, R. Preuss, and U. von Toussaint, editors, *Bayesian inference and maximum entropy methods in science and engineering*, AIP Conference Proceedings 735, pages 395–405, 2004.

[2] Andrew Gelman and Xiao-Li Meng. Simulating normalizing constants: from importance sampling to bridge sampling to path sampling. *Statist. Sci.*, 13(2):163–185, 1998.

[3] Matthew J. Beal and Zoubin Ghahramani. The variational Bayesian EM algorithm for incomplete data: with application to scoring graphical model structures. *Bayesian Statistics*, 7:453–464, 2003.

[4] I. R. McDonald and K. Singer. Machine calculation of thermodynamic properties of a simple fluid at supercritical temperatures. *J. Chem. Phys.*, 47(11):4766–4772, 1967.

[5] David J.C. MacKay. *Information Theory, Inference, and Learning Algorithms*. CUP, 2003. www.inference.phy.cam.ac.uk/mackay/itila/.

[6] Robert G. Edwards and Alan D. Sokal. Generalization of the Fortuin-Kasteleyn-Swendsen-Wang representation and Monte Carlo algorithm. *Phys.Rev. D*, 38(6), 1988.

[7] C. M. Fortuin and P. W. Kasteleyn. On the random-cluster model. I. Introduction and relation to other models. *Physica*, 57:536–564, 1972.

[8] R. H. Swendsen and J. S. Wang. Nonuniversal critical dynamics in Monte Carlo simulations. *Phys. Rev. Lett.*, 58(2):86–88, January 1987.

[9] Radford M. Neal. Annealed importance sampling. *Statistics and Computing*, 11:125–139, 2001.

[10] Charles H. Bennett. Efficient estimation of free energy differences from Monte Carlo data. *Journal of Computational Physics*, 22(2):245–268, October 1976.

[11] Vivek K. Gore and Mark R. Jerrum. The Swendsen-Wang process does not always mix rapidly. In *29th ACM Symposium on Theory of Computing*, pages 674–681, 1997.

[12] Bernd A. Berg and Thomas Neuhaus. Multicanonical ensemble: A new approach to simulate first-order phase transitions. *Phys. Rev. Lett.*, 68(1):9–12, January 1992.
